# Gaussian Processes in Reinforcement Learning

**Carl Edward Rasmussen** and **Malte Kuss**
Max Planck Institute for Biological Cybernetics
Spemannstraße 38, 72076 Tübingen, Germany
`{carl,malte.kuss}@tuebingen.mpg.de`

## Abstract

We exploit some useful properties of Gaussian process (GP) regression
models for reinforcement learning in continuous state spaces and dis-
crete time. We demonstrate how the GP model allows evaluation of the
value function in closed form. The resulting policy iteration algorithm is
demonstrated on a simple problem with a two dimensional state space.
Further, we speculate that the intrinsic ability of GP models to charac-
terise distributions of functions would allow the method to capture entire
distributions over future values instead of merely their expectation, which
has traditionally been the focus of much of reinforcement learning.

## 1 Introduction

Model-based control of discrete-time non-linear dynamical systems is typically exacer-
bated by the existence of multiple relevant time scales: a short time scale (the sampling
time) on which the controller makes decisions and where the dynamics are simple enough
to be conveniently captured by a model learning from observations, and a longer time scale
which captures the long-term consequences of control actions. For most non-trivial (non-
minimum phase) control tasks a policy relying solely on short time rewards will fail.

In reinforcement learning this problem is explicitly recognized by the distinction between
short-term (reward) and long-term (value) desiderata. The consistency between short- and
long-term goals are expressed by the Bellman equation, for discrete states $\mathbf{s}$ and actions $\mathbf{a}$:

$$V^\pi(\mathbf{s}) = \sum_{\mathbf{a}} \pi(\mathbf{s}, \mathbf{a}) \sum_{\mathbf{s}'} \mathcal{P}^{\mathbf{a}}_{\mathbf{s}, \mathbf{s}'} \left[ \mathcal{R}^{\mathbf{a}}_{\mathbf{s}, \mathbf{s}'} + \gamma V^\pi(\mathbf{s}') \right] \tag{1}$$

where $V^\pi(\mathbf{s})$ is the value (the expected long term reward) of state $\mathbf{s}$ while following policy
$\pi(\mathbf{s}, \mathbf{a})$, which is the probability of taking action $\mathbf{a}$ in state $\mathbf{s}$, and $\mathcal{P}^{\mathbf{a}}_{\mathbf{s}, \mathbf{s}'}$ is the transition
probability of going to state $\mathbf{s}'$ when applying action $\mathbf{a}$ given that we are in state $\mathbf{s}$, $\mathcal{R}^{\mathbf{a}}_{\mathbf{s}, \mathbf{s}'}$
denotes the immediate expected reward and $0 < \gamma < 1$ is the discount factor (see Sutton and
Barto (1998) for a thorough review). The Bellman equations are either solved iteratively by
policy evaluation, or alternatively solved directly (the equations are linear) and commonly
interleaved with policy improvement steps (policy iteration).

While the concept of a value function is ubiquitous in reinforcement learning, this is not
the case in the control community. Some non-linear model-based control is restricted to the
easier minimum-phase systems. Alternatively, longer-term predictions can be achieved by
concatenating short-term predictions, an approach which is made difficult by the fact that

uncertainty in predictions typically grows (precluding approaches based on the *certainty equivalence principle*) as the time horizon lengthens. See Quiñonero-Candela et al. (2003) for a full probabilistic approach based on Gaussian processes; however, implementing a controller based on this approach requires numerically solving multivariate optimisation problems for every control action. In contrast, having access to a value function makes computation of control actions much easier.

Much previous work has involved the use of function approximation techniques to represent the value function. In this paper, we exploit a number of useful properties of Gaussian process models for this purpose. This approach can be naturally applied in discrete time, continuous state space systems. This avoids the tedious discretisation of state spaces often required by other methods, eg. Moore and Atkeson (1995). In Dietterich and Wang (2002) kernel based methods (support vector regression) were also applied to learning of the value function, but in discrete state spaces.

In the current paper we use Gaussian process (GP) models for two distinct purposes: first to model the dynamics of the system (actually, we use one GP per dimension of the state space) which we will refer to as the *dynamics* GP and secondly the *value* GP for representing the value function. When computing the values, we explicitly take the uncertainties from the dynamics GP into account, and using the linearity of the GP, we are able to solve directly for the value function, avoiding slow policy evaluation iterations.

Experiments on a simple problem illustrates the viability of the method. For these experiments we use a greedy policy wrt. the value function. However, since our representation of the value function is stochastic, we could represent uncertainty about values enabling a principled attack of the exploration vs. exploitation tradeoff, such as in Bayesian Q-learning as proposed by Dearden et al. (1998). This potential is outlined in the discussion section.

## 2 Gaussian Processes and Value Functions

In a continuous state space we straight-forwardly generalize the Bellman equation (1) by substituting sums with integrals; further, we assume for simplicity of exposition that the policy is deterministic (see Section 4 for a further discussion):

$$V^{\pi}(\mathbf{s}) = \int \mathcal{P}_{\mathbf{s},\mathbf{s}'}^{\pi(\mathbf{s})} \big[ \mathcal{R}_{\mathbf{s},\mathbf{s}'}^{\pi(\mathbf{s})} + \gamma V^{\pi}(\mathbf{s}') \big] d\mathbf{s}' \qquad (2)$$

$$= \int \mathcal{P}_{\mathbf{s},\mathbf{s}'}^{\pi(\mathbf{s})} \mathcal{R}_{\mathbf{s},\mathbf{s}'}^{\pi(\mathbf{s})} d\mathbf{s}' + \gamma \int \mathcal{P}_{\mathbf{s},\mathbf{s}'}^{\pi(\mathbf{s})} V^{\pi}(\mathbf{s}') d\mathbf{s}'. \qquad (3)$$

This involves two integrals over the distribution of consecutive states $\mathbf{s}'$ visited when following the policy $\pi$. The transition probabilities $\mathcal{P}_{\mathbf{s},\mathbf{s}'}^{\pi}$ may include two sources of stochasticity: uncertainty in the model of the dynamics and stochasticity in the dynamics itself.

### 2.1 Gaussian Process Regression Models

In GP models we put a prior directly on functions and condition on observations to make predictions (see Williams and Rasmussen (1996) for details). The noisy targets $y_i = f(x_i) + \varepsilon_i$ are assumed jointly Gaussian with covariance function $k$:

$$\mathbf{y}|\mathbf{x} \sim \mathcal{N}(\mathbf{0}, \mathbf{K}), \quad \text{where } \mathbf{K}_{pq} = k(x_p, x_q). \qquad (4)$$

Throughout the remainder of this paper we use a Gaussian covariance function:

$$k(x_p, x_q|\theta) = v^2 \exp\big(-(x_p - x_q)^\top \mathbf{\Lambda}^{-1}(x_p - x_q)/2\big) + \delta_{pq}\sigma_n^2, \qquad (5)$$

where the positive elements of the diagonal matrix $\mathbf{\Lambda}$, $v$ and $\sigma_n^2$ are hyperparameters collected in $\theta$. The hyperparameters are fit by maximising the marginal likelihood (see again Williams and Rasmussen (1996)) using conjugate gradients.

The predictive distribution for a novel test input $x^*$ is Gaussian:

$$y^*|x^*, \mathbf{x}, \mathbf{y}, \theta \sim \mathcal{N}\big(\mu = \mathbf{k}(x^*, \mathbf{x})\mathbf{K}^{-1}\mathbf{y}, \ \sigma^2 = k(x^*, x^*) - \mathbf{k}(x^*, \mathbf{x})\mathbf{K}^{-1}\mathbf{k}(\mathbf{x}, x^*)\big). \quad (6)$$

## 2.2 Model Identification of System Dynamics

Given a set of $D$-dimensional observations of the form $(\mathbf{s}, \mathbf{a}, \mathbf{s}')$, we use a separate Gaussian process model for predicting each coordinate of the system dynamics. The inputs to each model are the state and action pair $x = (\mathbf{s}, \mathbf{a})$, the output is a (Gaussian) distribution over the consecutive state variable, $y = s'_d, d = 1, \ldots, D$ using eq. (6). Combining the predictive models we obtain a multivariate Gaussian distribution over the consecutive state: the transition probabilities $\mathcal{P}^{\pi}_{\mathbf{s}, \mathbf{s}'}$.

## 2.3 Policy Evaluation

We now turn towards the problem of evaluating $V(\mathbf{s})$ for a given policy $\pi$ over the continuous state space. In *policy evaluation* the Bellman equations are used as update rules. In order to apply this approach in the continuous case, we have to solve the two integrals in eq. (3).

For simple (eg. polynomial or Gaussian) reward functions $\mathcal{R}$ we can directly compute[1] the first Gaussian integral of eq. (3). Thus, the expected immediate reward, from state $\mathbf{s}_i$, following $\pi$ is:

$$\mathbf{R}_i = \int \mathcal{P}^{\pi(\mathbf{s}_i)}_{\mathbf{s}_i, \mathbf{s}'} \mathcal{R}^{\pi(\mathbf{s}_i)}_{\mathbf{s}_i, \mathbf{s}'} d\mathbf{s}', \ \text{ where } \ \mathcal{P}^{\pi}_{\mathbf{s}_i, \mathbf{s}'} = \mathcal{N}\big(\mu_i, \boldsymbol{\Sigma}_i = \mathrm{diag}(\sigma_1^2, \ldots, \sigma_D^2)\big), \quad (7)$$

in which the mean and covariance for the consecutive state are coordinate-wise given by eq. (6) evaluated on the dynamics GP.

The second integral of eq. (3) involves an expectation over the value function, which is modeled by the value GP as a function of the states. We need access to the value function at every point in the continuous state space, but we only explicitly represent values at a finite number of *support points*, $\mathbf{S} = \{\mathbf{s}_1, \ldots, \mathbf{s}_m\}$ and let the GP generalise to the entire space. Here we use the *mean* of the GP to represent the value[2] – see section 4 for an elaboration. Thus, we need to average the values over the distribution predicted for $\mathbf{s}'$. For a Gaussian covariance function[3] this can be done in closed form as shown by Girard et al. (2002). In detail, the Bellman equation for the value at support point $\mathbf{s}_i$ is:

$$\mathbf{V}_i = \mathbf{R}_i + \gamma \int \mathcal{P}^{\pi}_{\mathbf{s}_i, \mathbf{s}'} V(\mathbf{s}') d\mathbf{s}' = \mathbf{R}_i + \gamma \mathbf{W}_i \mathbf{K}_v^{-1} \mathbf{V}, \ \text{ where } \ \mathcal{P}^{\pi}_{\mathbf{s}_i, \mathbf{s}'} = \mathcal{N}(\mu_i, \boldsymbol{\Sigma}_i) \quad (8)$$

$$\text{and } \ \mathbf{W}_{ij} = |\boldsymbol{\Lambda}^{-1}\boldsymbol{\Sigma}_i + \mathbf{I}|^{-1/2} v^2 \exp\big(-(\mathbf{s}_j - \mu_i)^\top (\boldsymbol{\Lambda} + \boldsymbol{\Sigma}_i)^{-1}(\mathbf{s}_j - \mu_i)/2\big), \quad (9)$$

where $\mathbf{K}_v$ denotes the covariance matrix of the value GP, $\mathbf{W}_i$ is the $i$'th row of the $\mathbf{W}$ matrix and boldface $\mathbf{V}$ is the vector of values at the support points: $\mathbf{V} = (V(\mathbf{s}_1), \ldots, V(\mathbf{s}_m))^\top$. Note, that this equation implies a consistency between the value at the support points with the values at all other points. Equation (8) could be used for iterative policy evaluation. Notice however, that eq. (8) is a set of $m$ *linear* simultaneous equations in $\mathbf{V}$, which we can solve[4] explicitly:

$$\mathbf{V} = \mathbf{R} + \gamma \mathbf{W} \mathbf{K}_v^{-1} \mathbf{V} \implies \mathbf{V} = \big(\mathbf{I} - \gamma \mathbf{W} \mathbf{K}_v^{-1}\big)^{-1} \mathbf{R}. \quad (10)$$

The computational cost of solving this system is $\mathcal{O}(m^3)$, which is no more expensive than doing iterative policy evaluation, and equal to the cost of value GP prediction.

## 2.4 Policy Improvement

Above we demonstrated how to compute the value function for a given policy $\pi$. Now given a value function we can act greedily, thereby defining an implicit policy:

$$\pi(\mathbf{s}) \leftarrow \underset{\mathbf{a} \in \mathcal{A}(\mathbf{s})}{\operatorname{argmax}} \int \mathcal{P}_{\mathbf{s},\mathbf{s}'}^{\mathbf{a}} \left[ \mathcal{R}_{\mathbf{s},\mathbf{s}'}^{\mathbf{a}} + \gamma V(\mathbf{s}') \right] d\mathbf{s}', \qquad (11)$$

giving rise to $m$ one-dimensional optimisation problems (when the possible actions $a$ are scalar). As above we can solve the relevant integrals and in addition compute derivatives wrt. the action. Note also that application-specific constraints can often be reformulated as constraints in the above optimisation problem.

## 2.5 The Policy Iteration Algorithm

We now combine *policy evaluation* and *policy improvement* into *policy iteration* in which both steps alternate until a stable configuration is reached[5]. Thus given observations of system dynamics and a reward function we can compute a continuous value function and thereby an implicitly defined policy.

---

**Algorithm 1** Policy iteration, batch version

---

**1. Given:** $n$ observations of system dynamics of the form $(\mathbf{s}, a, \mathbf{s}')$ for a fixed time interval $\Delta t$, discount factor $\gamma$ and reward function $\mathcal{R}$

**2. Model Identification:** Model the system dynamics by Gaussian processes for each state coordinate and combine them to obtain a model $\mathcal{P}_{\mathbf{s},\mathbf{s}'}^{\pi} = \mathcal{N}(\mu_{\mathbf{s}'}, \boldsymbol{\Sigma}_{\mathbf{s}'})$

**3. Initialise Value Function:** Choose a set $\mathbf{S} = \{\mathbf{s}_1, \ldots, \mathbf{s}_m\}$ of $m$ support points and initialize $\mathbf{V}_i \leftarrow \mathcal{R}(\mathbf{s}_i)$. Fit Gaussian process hyperparameters for representing $V(\mathbf{s})$ using conjugate gradient optimisation of the marginal likelihood and set $\sigma_n^2$ to a small positive constant.

**4. Policy Iteration:**

**repeat**

    **for all** $\mathbf{s}_i \in \mathbf{S}$ **do**

        Find action $\mathbf{a}_i$ by solving equation (11) subject to problem specific constraints.

        Compute $\mathcal{P}_{\mathbf{s}_1,\mathbf{s}'_i}^{a_i}$ using the dynamics Gaussian processes

        Solve equation (7) in order to obtain $\mathbf{R}_i$

        Compute $i$'th row of $\mathbf{W}$ as in equation (9)

    **end for**

    $\mathbf{V} \leftarrow (\mathbf{I} - \gamma \mathbf{W} \mathbf{K}_v^{-1})^{-1} \mathbf{R}$

    Update Gaussian process hyperparameter for representing $V(\mathbf{s})$ to fit the new $\mathbf{V}$

**until** stabilisation of $\mathbf{V}$

---

The selection of the support points remains to be determined. When using the algorithm in an online setting, support points could naturally be chosen as the states visited, possibly selecting the ones which conveyed most new information about the system. In the experimental section, for simplicity of exposition we consider only the batch case, and use simply a regular grid of support points.

We have assumed for simplicity that the reward function is deterministic and known, but it would not be too difficult to also use a (GP) model for the rewards; any model that allows

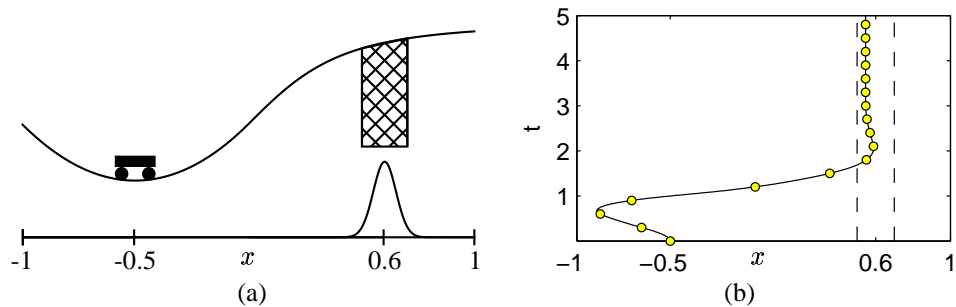

(a)    (b)

Figure 1: Figure (a) illustrates the mountain car problem. The car is initially standing motionless at $x = -0.5$ and the goal is to bring it up and hold it in the region $0.5 \leq x \leq 0.7$ such that $-0.1 \leq \dot{x} \leq 0.1$. The hatched area marks the target region and below the approximation by a Gaussian is shown (both projected onto the $x$ axis). Figure (b) shows the position $x$ of the car is when controlled according to (11) using the approximated value function after 6 policy improvements shown in Figure 3. The car reaches the target region in about five time steps but does not end up exactly at $x = 0.6$ due to uncertainty in the dynamics model. The circles mark the $\Delta t = 0.3$ second time steps.

evaluation of eq. (7) could be used. Similarly the greedy policy has been assumed, but generalisation to stochastic policies would not be difficult.

## 3    Illustrative Example

For reasons of presentability of the value function, we below consider the well-known mountain car problem "park on the hill", as described by Moore and Atkeson (1995) where the state-space is only two-dimensional. The setting depicted in Figure 1(a) consists of a frictionless, point-like, unit mass car on a hilly landscape described by

$$H(x) = \begin{cases} x^2 + x & \text{for } x < 0, \\ \frac{x}{\sqrt{1+5x^2}} & \text{for } x \geq 0. \end{cases} \quad (12)$$

The state of the system $\mathbf{s} = (x, \dot{x})$ is described by the position of the car and its speed which are constrained to $-1 \leq x \leq 1$ and $-2 \leq \dot{x} \leq 2$ respectively. As action a horizontal force $F$ in the range $-4 \leq F \leq 4$ can be applied in order to bring the car up into the target region which is a rectangle in state space such that $0.5 \leq x \leq 0.7$ and $-0.1 \leq \dot{x} \leq 0.1$. Note that the admissible range of forces is not sufficient to drive up the car greedily from the initial state $\mathbf{s}_0 = (-0.5, 0)$ such that a strategy has to be found which utilises the landscape in order to accelerate up the slope, which gives the problem its non-minimum phase character.

For system identification we draw samples $\{(\mathbf{s}_i, F_i)|i = 1, \ldots, n = 50\}$ uniformly from their respective admissible regions and simulate time steps of $\Delta t = 0.3$ seconds[6] forward in time using an ODE solver in order to get the consecutive states $\mathbf{s}_i'$. We then use two Gaussian processes to build a model to predict the system behavior from these examples for the two state variables independently using covariance functions of type eq. (5). Based

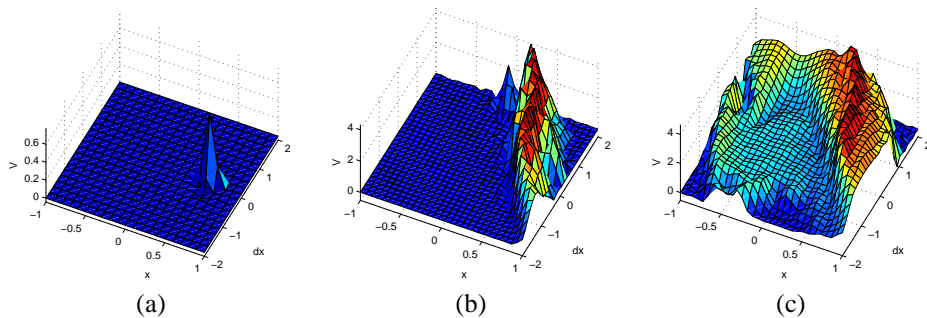

(a)                              (b)                              (c)

Figure 2: Figures (a-c) show the estimated value function for the mountain car example after initialisation (a), after the first iteration over $\mathbf{S}$ (b) and a nearly stabilised value function after 3 iterations (c). See also Figure 3 for the final value function and the corresponding state transition diagram

on 50 random examples, the relations can already be approximated to within root mean squared errors (estimated on 1000 test samples and considering the mean of the predicted distribution) of 0.02 for predicting $x$ and 0.2 for predicting $\dot{x}$.

Having a model of the system dynamics, the other necessary element to provide to the proposed algorithm is a reward function. In the formulation by Moore and Atkeson (1995) the reward is equal to 1 if the car is in the target region and 0 elsewhere. For convenience we approximate this cube by a Gaussian proportional to $\mathcal{N}([0.6, 0]', 0.05^2\mathbf{I})$ with maximum reward 1 as indicated in Figure 1(a). We now can solve the update equation (10) and also evaluate its gradient with respect to $F$. This enables us to efficiently solve the optimization problem eq. (11) subject to the constraints on $x$, $\dot{x}$ and $F$ described above. States outside the feasible region are assigned zero value and reward.

As support points for the value function we simply put a regular $21 \times 21$ grid onto the state-space and initialise the value function with the immediate rewards for these states, Figure 2(a). The standard deviation of the noise of the value GP representing $V(\mathbf{s})$ is set to $\sigma_v = 0.01$, and the discount factor to $\gamma = 0.8$. Following the policy iteration algorithm we estimate the value of all support points following the implicit policy (11) wrt. the initial value function, Figure 2(a). We then evaluate this policy and obtain an updated value function shown in Figure 2(b) where all points which can expect to reach the reward region in one time step gain value. If we iterate this procedure two times we obtain a value function as shown in Figure 2(c) in which the state space is already well organised. After five policy iterations the value function and therefore the implicit policy is stable, Figure 3(a). In Figure 3(b) a dynamic GP based state-transition diagram is shown, in which each support point $\mathbf{s}_i$ is connected to its predicted (mean) consecutive state $\mathbf{s}'_i$ when following the implicit policy. For some of the support points the model correctly predicts that the car will leave the feasible region, no matter what $|F| \leq 4$ is applied, which corresponds to areas with zero value in Figure 3(a).

If we control the car from $\mathbf{s}_0 = (-0.5, 0)$ according to the found policy the car gathers momentum by first accelerating left before driving up into the target region where it is balanced as illustrated in Figure 1(b). It shows that the 50 random examples of the system dynamics are sufficient for this task. The control policy found is probably very close to the optimally achievable.

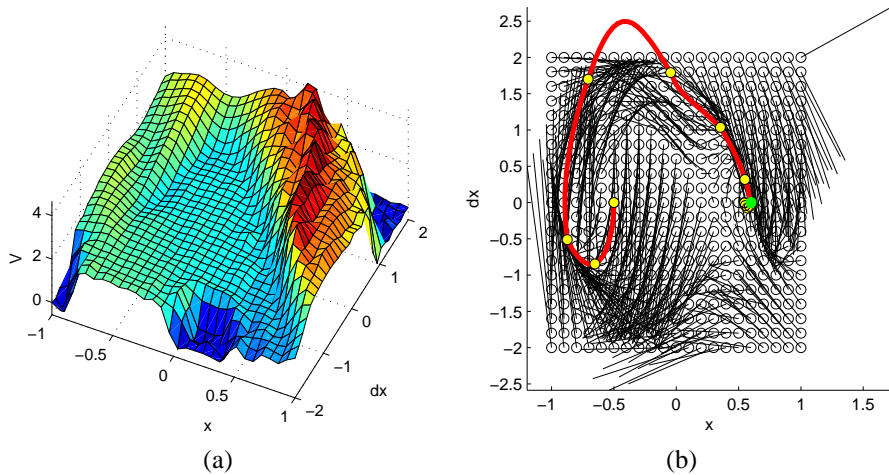

(a)  (b)

Figure 3: Figures (a) shows the estimated value function after 6 policy improvements (subsequent to Figures 2(a-c)) over $\mathbf{S}$ where $\mathbf{V}$ has stabilised. Figure (b) is the corresponding state transition diagram illustrating the implicit policy on the support points. The black lines connect $\mathbf{s}_i$ and the respective $\mathbf{s}'_i$ estimated by the dynamics GP when following the implicit greedy policy with respect to (a). The thick line marks the trajectory of the car for the movement described in Figure 1(b) based on the physics of the system. Note that the temporary violation of the constraint $\dot{x} < 2$ remains unnoticed using time intervals of $\Delta t = 0.3$; to avoid this the constraints could be enforced continuously in the training set.

## 4   Perspectives and Conclusion

Commonly the value function is defined to be the *expected* (discounted) future reward. Conceptually however, there is more to values than their expectations. The distribution over future reward could have small or large variance and identical means, two fairly different situations, that are treated identically when only the value expectation is considered. It is clear however, that a principled approach to the exploitation vs. exploration tradeoff requires a more faithful representation of value, as was recently proposed in Bayesian Q-learning (Dearden et al. 1998), and see also Attias (2003). For example, the large variance case is more attractive for exploration than the small variance case.

The GP representation of value functions proposed here lends itself naturally to this more elaborate concept of value. The GP model inherently maintains a full distribution over values, although in the present paper we have only used its expectation. Implementation of this would require a second set of Bellman-like equations for the second moment of the values at the support points. These equations would simply express consistency of uncertainty: the uncertainty of a value should be consistent with the uncertainty when following the policy. The values at the support points would be (Gaussian) distributions with individual variances, which is readily handled by using a full diagonal noise term in the place of $\delta_{pq}\sigma_n^2$ in eq. (5). The individual second moments can be computed in closed form (see derivations in Quiñonero-Candela et al. (2003)). However, iteration would be necessary to solve the combined system, as there would be no linearity corresponding to eq. (10) for the second moments. In the near future we will be exploring these possibilities.

Whereas only a batch version of the algorithm has been described here, it would obviously be interesting to explore its capabilities in an online setting, starting from scratch. This will require that we abandon the use of a greedy policy, to avoid risking to get stuck in a

local minima caused by an incomplete model of the dynamics. Instead, a stochastic policy should be used, which should not cause further computational problems as long as it is represented by a Gaussian (or perhaps more appropriately a mixture of Gaussians). A good policy should actively explore regions where we may gain a lot of information, requiring the notion of the value of information (Dearden et al. 1998). Since the information gain would come from a better dynamics GP model, it may not be an easy task in practice to optimise jointly information and value.

We have introduced Gaussian process models into continuous-state reinforcement learning tasks, to model the state dynamics and the value function. We believe that the good generalisation properties, and the simplicity of manipulation of GP models make them ideal candidates for these tasks. In a simple demonstration, our parameter-free algorithm converges rapidly to a good approximation of the value function.

Only the batch version of the algorithm was demonstrated. We believe that the full probabilistic nature of the transition model should facilitate the early states of an on-line process. Also, online addition of new observations in GP model can be done very efficiently. Only a simple problem was used, and it will be interesting to see how the algorithm performs on more realistic tasks. Direct implementation of GP models are suitable for up to a few thousand support points; in recent years a number of fast approximate GP algorithms have been developed, which could be used in more complex settings.

We are convinced that recent developments in powerful kernel-based probabilistic models for supervised learning such as GPs, will integrate well into reinforcement learning and control. Both the modeling and analytic properties make them excellent candidates for reinforcement learning tasks. We speculate that their fully probabilistic nature offers promising prospects for some fundamental problems of reinforcement learning.

### Acknowledgements

Both authors were supported by the German Research Council (DFG).

## Footnotes

[1] For more complex reward functions we may approximate it using eg. a Taylor expansion.

[2] Thus, here we are using the GP for noise free interpolation of the value function, and consequently set its noise parameter to a small positive constant (to avoid numerical problems)

[3] The covariance functions allowing analytical treatment in this way include Gaussian and polynomial, and mixtures of these.

[4] We conjecture that the matrix $\mathbf{I} - \gamma \mathbf{W} \mathbf{K}_v^{-1}$ is non-singular under mild conditions, but have not yet devised a formal proof.

[5]Assuming convergence, which we have not proven.

[6]Note that $\Delta t = 0.3$ seconds seems to be an order of magnitude slower than the time scale usually considered in the literature. Our algorithm works equally well for shorter time steps ($\gamma$ should be increased); for even longer time steps, modeling of the dynamics gets more complicated, and eventually for large enough $\Delta t$ control is no-longer possible.

### References

Attias, H. (2003). Planning by probabilistic inference. In *Proceedings of the Ninth International Workshop on Artificial Intelligence and Statistics*.

Dearden, R., N. Friedman, and S. J. Russell (1998). Bayesian Q-learning. In *Fifteenth National Conference on Artificial Intelligence (AAAI)*.

Dieterich, T. G. and X. Wang (2002). Batch value function approximation via support vectors. In *Advances in Neural Information Processing Systems 14*, Cambridge, MA, pp. 1491–1498. MIT Press.

Girard, A., C. E. Rasmussen, J. Qui˜nonero-Candela, and R. Murray-Smith (2002). Multiple-step ahead prediction for non linear dynamic systems – a Gaussian process treatment with propagation of the uncertainty. In *Advances in Neural Information Processing Systems 15*.

Moore, A. W. and C. G. Atkeson (1995). The parti-game algorithm for variable resolution reinforcement learning in multidimensional state-spaces. *Machine Learning 21*, 199–233.

Qui˜nonero-Candela, J., A. Girard, J. Larsen, and C. E. Rasmussen (2003). Propagation of uncertainty in Bayesian kernel models - application to multiple-step ahead forecasting. In *Proceedings of the 2003 IEEE Conference on Acoustics, Speech, and Signal Processing*.

Sutton, R. S. and A. G. Barto (1998). *Reinforcement Learning*. Cambridge, Massachusetts: MIT Press.

Williams, C. K. I. and C. E. Rasmussen (1996). Gaussian processes for regression. In *Advances in Neural Information Processing Systems 8*.
